# Nonparametric Model-Based Reinforcement Learning

**Christopher G. Atkeson**
College of Computing, Georgia Institute of Technology,
Atlanta, GA 30332-0280, USA
ATR Human Information Processing,
2-2 Hikaridai, Seiko-cho, Soraku-gun, 619-02 Kyoto, Japan
cga@cc.gatech.edu
http://www.cc.gatech.edu/fac/Chris.Atkeson/

## Abstract

This paper describes some of the interactions of model learning algorithms and planning algorithms we have found in exploring model-based reinforcement learning. The paper focuses on how local trajectory optimizers can be used effectively with learned nonparametric models. We find that trajectory planners that are fully consistent with the learned model often have difficulty finding reasonable plans in the early stages of learning. Trajectory planners that balance obeying the learned model with minimizing cost (or maximizing reward) often do better, even if the plan is not fully consistent with the learned model.

## 1 INTRODUCTION

We are exploring the use of nonparametric models in robot learning (Atkeson et al., 1997b; Atkeson and Schaal, 1997). This paper describes the interaction of model learning algorithms and planning algorithms, focusing on how local trajectory optimization can be used effectively with nonparametric models in reinforcement learning. We find that trajectory optimizers that are fully consistent with the learned model often have difficulty finding reasonable plans in the early stages of learning. The message of this paper is that a planner should not be entirely consistent with the learned model during model-based reinforcement learning. Trajectory optimizers that balance obeying the learned model with minimizing cost (or maximizing reward) often do better, even if the plan is not fully consistent with the learned model:

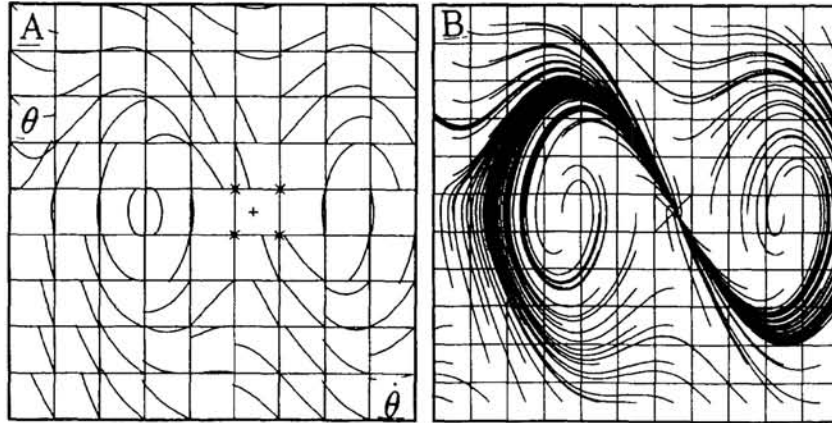

Figure 1: A: Planning in terms of trajectory segments. B: Planning in terms of trajectories all the way to a goal point.

Two kinds of reinforcement learning algorithms are direct (non-model-based) and indirect (model-based). Direct reinforcement learning algorithms learn a policy or value function without explicitly representing a model of the controlled system (Sutton et al., 1992). Model-based approaches learn an explicit model of the system simultaneously with a value function and policy (Sutton, 1990, 1991a,b; Barto et al., 1995; Kaelbling et al., 1996). We will focus on model-based reinforcement learning, in which the learner uses a planner to derive a policy from a learned model and an optimization criterion.

## 2  CONSISTENT LOCAL PLANNING

An efficient approach to dynamic programming, a form of global planning, is to use local trajectory optimizers (Atkeson, 1994). These local planners find a plan for each starting point in a grid in the state space. Figure 1 compares the output of a traditional cell based dynamic programming process with the output of a planner based on integrating local plans. Traditional dynamic programming generates trajectory segments from each cell to neighboring cells, while the planner we use generates entire trajectories. These locally optimal trajectories have local policies and local models of the value function along the trajectories (Dyer and McReynolds, 1970; Jacobson and Mayne, 1970). The locally optimal trajectories are made consistent with their neighbors by using the local value function to predict the value of a neighboring trajectory. If all the local value functions are consistent with their neighbors the aggregate value function is a unique solution to the Bellman equation and the corresponding trajectories and policy are globally optimal. We would like any local planning algorithm to produce a local model of the value function so we can perform this type of consistency checking. We would also like a local policy from the local planner, so we can respond to disturbances and modeling errors.

Differential dynamic programming is a local planner that has these characteristics (Dyer and McReynolds, 1970; Jacobson and Mayne, 1970). Differential dynamic programming maintains a local quadratic model of the value function along the current best trajectory $\mathbf{x}^*(t)$:

$$V(\mathbf{x},t) = V_0(t) + V_x(t)(\mathbf{x} - \mathbf{x}^*(t))^{\mathrm{T}} + 0.5(\mathbf{x} - \mathbf{x}^*(t))^{\mathrm{T}}V_{xx}(t)(\mathbf{x} - \mathbf{x}^*(t)) \quad (1)$$

as well as a local linear model of the corresponding policy:

$$\mathbf{u}(\mathbf{x}, t) = \mathbf{u}^*(t) + \mathbf{K}(t)(\mathbf{x} - \mathbf{x}^*(t)) \tag{2}$$

$\mathbf{u}(\mathbf{x}, t)$ is the local policy at time $t$, the control signal $\mathbf{u}$ as a function of state $\mathbf{x}$. $\mathbf{u}^*(t)$ is the model's estimate of the control signal necessary to follow the current best trajectory $\mathbf{x}^*(t)$. $\mathbf{K}(t)$ are the feedback gains that alter the control signals in response to deviations from the current best trajectory. These gains are also the first derivative of the policy along the current best trajectory.

The first phase of each optimization iteration is to apply the current local policy to the learned model, integrating the modeled dynamics forward in time and seeing where the simulated trajectory goes. The second phase of the differential dynamic programming approach is to calculate the components of the local quadratic model of the value function at each point along the trajectory: the constant term $V_0(t)$, the gradient $V_x(t)$, and the Hessian $V_{xx}(t)$. These terms are constructed by integrating backwards in time along the trajectory. The value function is used to produce a new policy, which is represented using a new $\mathbf{x}^*(t)$, $\mathbf{u}^*(t)$, and $\mathbf{K}(t)$.

The availability of a local value function and policy is an attractive feature of differential dynamic programming. However, we have found several problems when applying this method to model-based reinforcement learning with nonparametric models:

1. Methods that enforce consistency with the learned model need an initial trajectory that obeys that model, which is often difficult to produce.

2. The integration of the learned model forward in time often blows up when the learned model is inaccurate or when the plant is unstable and the current policy fails to stabilize it.

3. The backward integration to produce the value function and a corresponding policy uses derivatives of the learned model, which are often quite inaccurate in the early stages of learning, producing inaccurate value function estimates and ineffective policies.

## 3   INCONSISTENT LOCAL PLANNING

To avoid the problems of consistent local planners, we developed a trajectory optimization approach that does not integrate the learned model and does not require full consistency with the learned model. Unfortunately, the price of these modifications is that the method does not produce a value function or a policy, just a trajectory $(\mathbf{x}(t), \mathbf{u}(t))$. To allow inconsistency with the learned model, we represent the state history $\mathbf{x}(t)$ and the control history $\mathbf{u}(t)$ separately, rather than calculate $\mathbf{x}(t)$ from the learned model and $\mathbf{u}(t)$. We also modify the original optimization criterion $C = \sum_k c(\mathbf{x}_k, \mathbf{u}_k)$ by changing the hard constraint that $\mathbf{x}_{k+1} = \mathbf{f}(\mathbf{x}_k, \mathbf{u}_k)$ on each time step into a soft constraint:

$$C_{new} = \sum_k \left[ c(\mathbf{x}_k, \mathbf{u}_k) + \lambda |\mathbf{x}_{k+1} - \mathbf{f}(\mathbf{x}_k, \mathbf{u}_k)|^2 \right] \tag{3}$$

$c(\mathbf{x}_k, \mathbf{u}_k)$ is the one step cost in the original optimization criterion. $\lambda$ is the penalty on the trajectory being inconsistent with the learned model $\widehat{\mathbf{x}}_{k+1} = \mathbf{f}(\mathbf{x}_k, \mathbf{u}_k)$. $|\mathbf{x}_{k+1} - \mathbf{f}(\mathbf{x}_k, \mathbf{u}_k)|$ is the magnitude of the mismatch of the trajectory and the model prediction at time step $k$ in the trajectory. $\lambda$ provides a way to control the amount of inconsistency. A small $\lambda$ reflects lack of confidence in the model, and allows

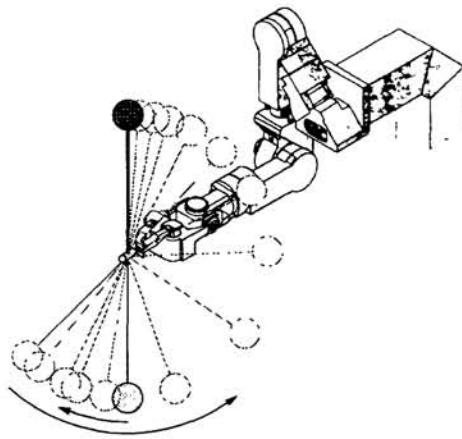

Figure 2: The SARCOS robot arm with a pendulum gripped in the hand. The pendulum axis is aligned with the fingers and with the forearm in this arm configuration.

the optimized trajectory to be inconsistent with the model in favor of reducing $c(\mathbf{x}_k, \mathbf{u}_k)$. A large $\lambda$ reflects confidence in the model, and forces the optimized trajectory to be more consistent with the model. $\lambda$ can increase with time or with the number of learning trials. If we use a model that estimates the confidence level of a prediction, we can vary $\lambda$ for each lookup based on $\mathbf{x}_k$ and $\mathbf{u}_k$. Locally weighted learning techniques provide exactly this type of local confidence estimate (Atkeson et al., 1997a).

Now that we are not integrating the trajectory we can use more compact representations of the trajectory, such as splines (Cohen, 1992) or wavelets (Liu et al., 1994). We no longer require that $\mathbf{x}_{k+1} = \mathbf{f}(\mathbf{x}_k, \mathbf{u}_k)$, which is a condition difficult to fulfill without having $\mathbf{x}$ and $\mathbf{u}$ represented as independent values on each time step. We can now parameterize the trajectory using the spline knot points, for example. In this work we used B splines (Cohen, 1992) to represent the trajectory. Other choices for spline basis functions would probably work just as well. We can use any nonlinear programming or function optimization method to minimize the criterion in Eq. 3. In this work we used Powell's method (Press et al., 1988) to optimize the knot points, a method which is convenient to use but not particularly efficient.

## 4   IMPLEMENTATION ON AN ACTUAL ROBOT

Both local planning methods work well with learned parametric models. However, differential dynamic programming did not work at all with learned nonparametric models, for reasons already discussed. This section describes how the inconsistent local planning method was used in an application of model-based reinforcement learning: robot learning from demonstration using a pendulum swing up task (Atkeson and Schaal, 1997). The pendulum swing up task is a more complex version of the pole or broom balancing task (Spong, 1995). The hand holds the axis of the pendulum, and the pendulum rotates about this hinge in an angular movement (Figure 2). Instead of starting with the pendulum vertical and above its rotational joint, the pendulum is hanging down from the hand, and the goal of the swing up task is to move the hand so that the pendulum swings up and is then balanced in the inverted position. The swing up task was chosen for study because it is a difficult dynamic maneuver and requires practice for humans to learn, but it is easy to tell if the task is successfully executed (at the end of the task the pendulum is balanced upright and does not fall down).

We implemented learning from demonstration on a hydraulic seven degree of free-

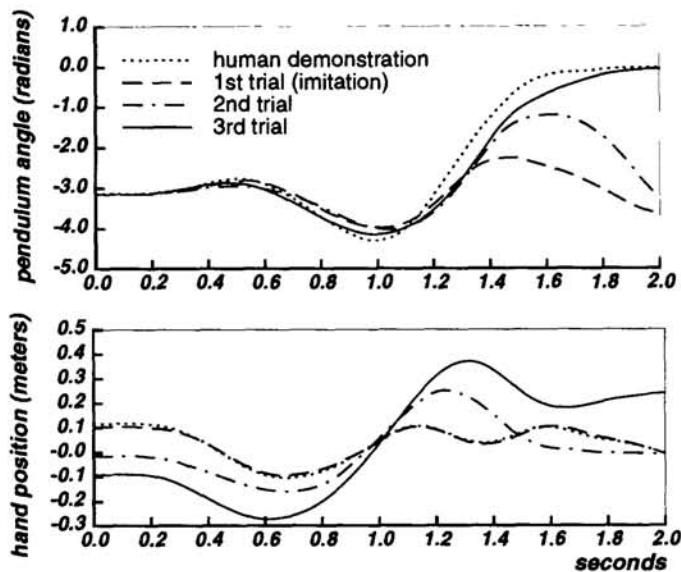

Figure 3: The hand and pendulum motion during robot learning from demonstration using a nonparametric model.

dom anthropomorphic robot arm (SARCOS Dextrous Arm located at ATR, Figure 2). The robot observed its own performance with the same stereo vision system that was used to observe the human demonstrations.

The robot observed a human swinging up a pendulum using a horizontal hand movement (dotted line in Figure 3). The most obvious approach to learning from demonstration is to have the robot imitate the human motion, by following the human hand trajectory. The dashed lines in Figures 3 show the robot hand motion as it attempts to follow the human demonstration of the swing up task, and the corresponding pendulum angles. Because of differences in the task dynamics for the human and for the robot, this direct imitation failed to swing the pendulum up, as the pendulum did not get even halfway up to the vertical position, and then oscillated about the hanging down position.

The approach we used was to apply a planner to finding a swing up trajectory that worked for the robot, based on learning both a model and a reward function and using the human demonstration to initialize the planning process. The data collected during the initial imitation trial and subsequent trials was used to build a model. Nonparametric models were constructed using locally weighted learning as described in (Atkeson et al., 1997a). These models did not use knowledge of the model structure but instead assumed a general relationship:

$$\dot{\theta}_{k+1} = model(\theta_k, \dot{\theta}_k, x_k, \dot{x}_k, \ddot{x}_k) \tag{4}$$

where $\theta$ is the pendulum angle and $x$ is the hand position. Training data from the demonstrations was stored in a database, and a local model was constructed to answer each query. Meta-parameters such as distance metrics were tuned using cross validation on the training set. For example, cross validation was able to quickly establish that hand position and velocity ($x$ and $\dot{x}$) played an insignificant role in predicting future pendulum angular velocities.

The planner used a cost function that penalizes deviations from the demonstration trajectory sampled at $60Hz$:

$$c(\mathbf{x}_k, \mathbf{u}_k) = (\mathbf{x}_k - \mathbf{x}_k^{\mathrm{d}})^{\mathrm{T}}(\mathbf{x}_k - \mathbf{x}_k^{\mathrm{d}}) + \mathbf{u}_k^{\mathrm{T}}\mathbf{u}_k \tag{5}$$

where the state is $\mathbf{x} = (\theta, \dot{\theta}, x, \dot{x})$, $\mathbf{x}^{\mathrm{d}}$ is the demonstrated motion, $k$ is the sample index, and the control is $\mathbf{u} = (\ddot{x})$. Equation 3 was optimized using B splines to represent $\mathbf{x}$ and $\mathbf{u}$. The knot points for $\mathbf{x}$ and $\mathbf{u}$ were initially separately optimized to minimize

$$(\mathbf{x}_k - \mathbf{x}_k^{\mathrm{d}})^{\mathrm{T}}(\mathbf{x}_k - \mathbf{x}_k^{\mathrm{d}}) \tag{6}$$

and

$$(\mathbf{u}_k - \mathbf{u}_k^{\mathrm{d}})^{\mathrm{T}}(\mathbf{u}_k - \mathbf{u}_k^{\mathrm{d}}) \tag{7}$$

The tolerated inconsistency, $\lambda$ was kept constant during a set of trials and set at values ranging from 100 to 100000. The exact value of $\lambda$ did not make much difference. Learning failed when $\lambda$ was set to zero, as there was no way for the learned model to affect the plan. The planning process failed when $\lambda$ was set too high, enforcing the learned model too strongly.

The next attempt got the pendulum up a little more. Adding this new data to the database and replanning resulted in a movement that succeeded (trial 3 in Figure 3). The behavior shown in Figure 3 is quite repeatable. The balancing behavior at the end of the trial is learned separately and continues for several minutes, at which point the trial is automatically terminated (Schaal, 1997).

## 5   DISCUSSION AND CONCLUSION

We applied locally weighted regression (Atkeson et al., 1997a) in an attempt to avoid the structural modeling errors of idealized parametric models during model-based reinforcement learning, and also to see if a priori knowledge of the structure of the task dynamics was necessary. In an exploration of the swingup task, we found that these nonparametric models required a planner that ignored the learned model to some extent. The fundamental reason for this is that planners amplify modeling error. Mechanisms for this amplification include:

- The planners take advantage of any modeling error to reduce the cost of the planned trajectory, so the planning process seeks out modeling error that reduces apparent cost.
- Some planners use derivatives of the model, which amplifies any noise in the model.

Models that support fast learning will have errors and noise. For example, in order to learn a model of the complexity necessary to accurately model the full robot dynamics between the commanded and actual hand accelerations a large amount of data is required, independent of modeling technique. The input would be 21 dimensional (robot state and command) ignoring actuator dynamics. Because there are few robot trials during learning, there is not enough data to make such a model even just in the vicinity of a successful trajectory. If it was required that enough data is collected during learning to make an accurate model, robot learning would be greatly slowed down.

One solution to this error amplification is to bias the nonparametric modeling tools to oversmooth the data. This reduces the benefit of nonparametric modeling, and also ignores the true learned model to some degree. Our solution to this problem is to introduce a controlled amount of inconsistency with the learned model into the planning process. The control parameter $\lambda$ is explicit and can be changed as a function of time, amount of data, or as a function of confidence in the model at the query point.

# References

Atkeson, C. G. (1994). Using local trajectory optimizers to speed up global optimization in dynamic programming. In Cowan, J. D., Tesauro, G., and Alspector, J., editors, *Advances in Neural Information Processing Systems 6*, pages 663–670. Morgan Kaufmann, San Mateo, CA.

Atkeson, C. G., Moore, A. W., and Schaal, S. (1997a). Locally weighted learning. *Artificial Intelligence Review*, 11:11–73.

Atkeson, C. G., Moore, A. W., and Schaal, S. (1997b). Locally weighted learning for control. *Artificial Intelligence Review*, 11:75–113.

Atkeson, C. G. and Schaal, S. (1997). Robot learning from demonstration. In *Proceedings of the 1997 International Conference on Machine Learning*.

Barto, A. G., Bradtke, S. J., and Singh, S. P. (1995). Learning to act using real-time dynamic programming. *Artificial Intelligence*, 72(1):81–138.

Cohen, M. F. (1992). Interactive spacetime control for animation. *Computer Graphics*, 26(2):293–302.

Dyer, P. and McReynolds, S. (1970). *The Computational Theory of Optimal Control*. Academic, NY.

Jacobson, D. and Mayne, D. (1970). *Differential Dynamic Programming*. Elsevier, NY.

Kaelbling, L. P., Littman, M. L., and Moore, A. W. (1996). Reinforcement learning: A survey. *Journal of Artificial Intelligence Research*, 4:237–285.

Liu, Z., Gortler, S. J., and Cohen, M. F. (1994). Hierarchical spacetime control. *Computer Graphics (SIGGRAPH '94 Proceedings)*, pages 35–42.

Press, W. H., Teukolsky, S. A., Vetterling, W. T., and Flannery, B. P. (1988). *Numerical Recipes in C*. Cambridge University Press, New York, NY.

Schaal, S. (1997). Learning from demonstration. In Mozer, M. C., Jordan, M., and Petsche, T., editors, *Advances in Neural Information Processing Systems 9*, pages 1040–1046. MIT Press, Cambridge, MA.

Spong, M. W. (1995). The swing up control problem for the acrobot. *IEEE Control Systems Magazine*, 15(1):49–55.

Sutton, R. S. (1990). Integrated architectures for learning, planning, and reacting based on approximating dynamic programming. In *Seventh International Machine Learning Workshop*, pages 216–224. Morgan Kaufmann, San Mateo, CA. http://envy.cs.umass.edu/People/sutton/publications.html.

Sutton, R. S. (1991a). Dyna, an integrated architecture for learning, planning and reacting. http://envy.cs.umass.edu/People/sutton/publications.html, Working Notes of the 1991 AAAI Spring Symposium on Integrated Intelligent Architectures pp. 151–155 and SIGART Bulletin 2, pp. 160-163.

Sutton, R. S. (1991b). Planning by incremental dynamic programming. In *Eighth International Machine Learning Workshop*, pages 353–357. Morgan Kaufmann, San Mateo, CA. http://envy.cs.umass.edu/People/sutton/publications.html.

Sutton, R. S., Barto, A. G., and Williams, R. J. (1992). Reinforcement learning is direct adaptive optimal control. *IEEE Control Systems Magazine*, 12:19—22.
